# Learning Invariant Representations of Molecules for Atomization Energy Prediction

**Grégoire Montavon**[1]*, **Katja Hansen**[2], **Siamac Fazli**[1], **Matthias Rupp**[3], **Franziska Biegler**[1], **Andreas Ziehe**[1], **Alexandre Tkatchenko**[2], **O. Anatole von Lilienfeld**[4], **Klaus-Robert Müller**[1,5]†

1. Machine Learning Group, TU Berlin
2. Fritz-Haber-Institut der Max-Planck-Gesellschaft, Berlin
3. Institute of Pharmaceutical Sciences, ETH Zurich
4. Argonne Leadership Computing Facility, Argonne National Laboratory, Lemont, IL
5. Dept. of Brain and Cognitive Engineering, Korea University

## Abstract

The accurate prediction of molecular energetics in chemical compound space is a crucial ingredient for rational compound design. The inherently graph-like, non-vectorial nature of molecular data gives rise to a unique and difficult machine learning problem. In this paper, we adopt a learning-from-scratch approach where quantum-mechanical molecular energies are predicted directly from the raw molecular geometry. The study suggests a benefit from setting flexible priors and enforcing invariance stochastically rather than structurally. Our results improve the state-of-the-art by a factor of almost three, bringing statistical methods one step closer to chemical accuracy.

## 1 Introduction

The accurate prediction of molecular energetics in chemical compound space (CCS) is a crucial ingredient for compound design efforts in chemical and pharmaceutical industries. One of the major challenges consists of making quantitative estimates in CCS at moderate computational cost (milliseconds per compound or faster). Currently only high level quantum-chemistry calculations, which can take days per molecule depending on property and system, yield the desired "chemical accuracy" of 1 kcal/mol required for computational molecular design.

This problem has only recently captured the interest of the machine learning community (Baldi et al., 2011). The inherently graph-like, non-vectorial nature of molecular data gives rise to a unique and difficult machine learning problem. A central question is how to represent molecules in a way that makes prediction of molecular properties feasible and accurate (Von Lilienfeld and Tuckerman, 2006). This question has already been extensively discussed in the cheminformatics literature, and many so-called molecular descriptors exist (Todeschini and Consonni, 2009). Unfortunately, they often require a substantial amount of domain knowledge and engineering. Furthermore, they are not necessarily transferable across the whole chemical compound space.

In this paper, we pursue a more direct approach initiated by Rupp et al. (2012) to the problem. We learn the mapping between the molecule and its atomization energy from scratch[1] using the "Coulomb matrix" as a low-level molecular descriptor (Rupp et al., 2012). As we will see later, an

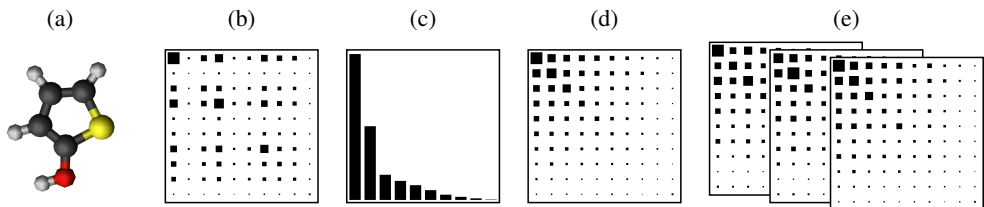

Figure 1: Different representations of the same molecule: (a) raw molecule with Cartesian coordinates and associated charges, (b) original (non-sorted) Coulomb matrix as computed by Equation 1, (c) eigenspectrum of the Coulomb matrix, (d) sorted Coulomb matrix, (e) set of randomly sorted Coulomb matrices.

inherent problem of the Coulomb matrix descriptor is that it lacks invariance with respect to permutation of atom indices, thus, leading to an exponential blow-up of the problem's dimensionality. We center the discussion around the two following questions: How to inject permutation invariance optimally into the machine learning model? What are the model characteristics that lead to the highest prediction accuracy?

Our study extends the work of Rupp et al. (2012) by empirically comparing several methods for enforcing permutation invariance: (1) computing the sorted eigenspectrum of the Coulomb matrix, (2) sorting the rows and columns by their respective norm and (3), a new idea, *randomly* sorting rows and columns in order to associate a set of randomly sorted Coulomb matrices to each molecule, thus extending the dataset considerably. These three representations are then compared in the light of several models such as Gaussian kernel ridge regression or multilayer neural networks where the Gaussian prior is traded against more flexibility and the ability to learn the representation directly from the data.

**Related Work**

In atomic-scale physics and in material sciences, neural networks have been used to model the potential energy surface of single systems (e.g., the dynamics of a single molecule over time) since the early 1990s (Lorenz et al., 2004; Manzhos and Carrington, 2006; Behler, 2011). Recently, Gaussian processes were used for this as well (Bartók et al., 2010). The major difference to the problem presented here is that previous work in modeling quantum mechanical energies looked mostly at the dynamics of one molecule, whereas we use data from different molecules simultaneously ("learning across chemical compound space"). Attempts in this direction have been rare (Balabin and Lomakina, 2009; Hautier et al., 2010; Balabin and Lomakina, 2011).

## 2   Representing Molecules

Electronic structure methods based on quantum-mechanical *first principles*, only require a set of nuclear charges $Z_i$ and the corresponding Cartesian coordinates of the atomic positions in 3D space $\mathbf{R_i}$ as an input for the calculation of molecular energetics. Here we use exactly the same information as input for our machine learning algorithms. Specifically, for each molecule, we construct the so-called Coulomb matrix $C$, that contains information about $Z_i$ and $\mathbf{R_i}$ in a way that preserves many of the required properties of a good descriptor (Rupp et al., 2012):

$$C_{ij} = \begin{cases} 0.5 Z_i^{2.4} & \forall i = j \\ \frac{Z_i Z_j}{|\mathbf{R}_i - \mathbf{R}_j|} & \forall i \neq j. \end{cases} \tag{1}$$

The diagonal elements of the Coulomb matrix correspond to a polynomial fit of the potential energies of the free atoms, while the off-diagonal elements encode the Coulomb repulsion between all possible pairs of nuclei in the molecule. As such, the Coulomb matrix is invariant to translations and rotations of the molecule in 3D space; both transformations must keep the potential energy of the molecule constant by definition.

Two problems with the Coulomb matrix representation that prevent it from being used out-of-the-box in a vector-space model are the following: (1) the dimension of the Coulomb matrix depends

on the number of atoms in the molecule and (2) the ordering of atoms in the Coulomb matrix is undefined, that is, many Coulomb matrices can be associated to the same molecule by just permuting rows and columns.

The first problem can be mitigated by introducing "invisible atoms" in the molecules, that have nuclear charge zero and do not interact with other atoms. These invisible atoms do not influence the physics of the molecule of interest and make the total number of atoms in the molecule sum to a constant $d$. In practice, this corresponds to padding the Coulomb matrix by zero-valued entries so that the Coulomb matrix has size $d \times d$, as it has been done by Rupp et al. (2012).

Solving the second problem is more difficult and has no obvious physically plausible workaround. Three candidate representations are depicted in Figure 1 and presented below.

## 2.1 Eigenspectrum Representation

The eigenspectrum representation (Rupp et al., 2012) is obtained by solving the eigenvalue problem $Cv = \lambda v$ under the constraint $\lambda_i \geq \lambda_{i+1}$ where $\lambda_i > 0$. The spectrum $(\lambda_1, \ldots, \lambda_d)$ is used as the representation. It is easy to see that this representation is invariant to permutation of atoms in the Coulomb matrix.

On the other hand, the dimensionality of the eigenspectrum $d$ is low compared to the initial $3d - 6$ degrees of freedom of most molecules. While this sharp dimensionality reduction may yield some useful built-in regularization, it may also introduce unrecoverable noise.

## 2.2 Sorted Coulomb Matrices

Another solution to the ordering problem is to choose the permutation of atoms whose associated Coulomb matrix $C$ satisfies $||C_i|| \geq ||C_{i+1}|| \; \forall \; i$ where $C_i$ denotes the $i^{\text{th}}$ row of the Coulomb matrix. Unlike the eigenspectrum representation, two different molecules have necessarily different associated sorted Coulomb matrices.

## 2.3 Random(-ly sorted) Coulomb Matrices

A way to deal with the larger dimensionality subsequent to taking the whole Coulomb matrix instead of the eigenspectrum is to extend the dataset with Coulomb matrices that are randomly sorted. This is achieved by associating a conditional distribution over Coulomb matrices $p(C|M)$ to each molecule $M$. Let $\mathcal{C}(M)$ define the set of matrices that are valid Coulomb matrices of the molecule $M$. The unnormalized probability distribution from which we would like to sample Coulomb matrices is defined as:

$$p^\star(C|M) = \sum_n 1_{C \in \mathcal{C}(M)} \cdot 1_{\{||C_i|| + n_i \geq ||C_{i+1}|| + n_{i+1} \forall i\}} \cdot p_{\mathcal{N}(0, \sigma I)}(n) \qquad (2)$$

The first term constrains the sample to be a valid Coulomb matrix of M, the second term ensures the sorting constraint and the third term defines the randomness parameterized by the noise level $\sigma$. Sampling from this distribution can be achieved approximately using the following algorithm:

---
**Algorithm for generating a random Coulomb matrix**

1. Take any Coulomb matrix $C$ among the set of matrices that are valid Coulomb matrices of $M$ and compute its row norm $||C|| = (||C_1||, \ldots, ||C_d||)$.

2. Draw $n \sim \mathcal{N}(0, \sigma I)$ and find the permutation $P$ that sorts $||C|| + n$, that is, find the permutation that satisfies $\text{permute}_P(||C|| + n) = \text{sort}(||C|| + n)$.

3. Permute $C$ row-wise and then column-wise with the same permutation, that is, $C_{\text{random}} = \text{permutecols}_P(\text{permuterows}_P(C))$.

---

The idea of dataset extension has already been used in the context of handwritten character recognition by, among others, LeCun et al. (1998), Ciresan et al. (2010) and in the context of support vector machines, by DeCoste and Schölkopf (2002). Random Coulomb matrices can be used at

Input (Coulomb matrix)                                                          Output (atomization energy)

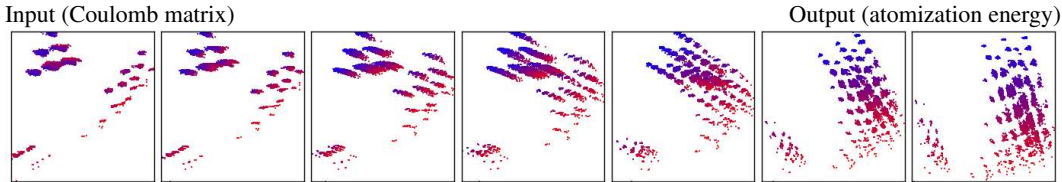

Figure 2: Two-dimensional PCA of the data with increasingly strong label contribution (from left to right). Molecules with low atomization energies are depicted in red and molecules with high atomization energies are depicted in blue. The plots suggest an interesting mix of global and local statistics with highly non-Gaussian distributions.

training time in order to multiply the number of data points but *also* at prediction time: predicting the property of a molecule consists of predicting the properties for all Coulomb matrices among the distribution of Coulomb matrices associated to $M$ and output the average of all these predictions $y = \mathbb{E}_{C|M}[f(C)]$.

## 3  Predicting Atomization Energies

The atomization energy $E$ quantifies the potential energy stored in all chemical bonds. As such, it is defined as the difference between the potential energy of a molecule and the sum of potential energies of its composing isolated atoms. The potential energy of a molecule is the solution to the electronic Schrödinger equation $H\Phi = E\Phi$, where $H$ is the Hamiltonian of the molecule and $\Phi$ is the state of the system. Note that the Hamiltonian is uniquely defined by the Coulomb matrix up to rotation and translation symmetries. A dataset $\{(M_1, E_1), \ldots, (M_n, E_n)\}$ is created by running a Schrödinger equation solver on a small set of molecules. Figure 2 shows a two-dimensional PCA visualization of the dataset where input and output distributions exhibit an interesting mix of local and global statistics.

Obtaining atomization energies from the Schrödinger equation solver is computationally expensive and, as a consequence, only a fraction of the molecules in the chemical compound space can be labeled. The learning algorithm is then asked to generalize from these few data points to unseen molecules. In this section, we show how two algorithms of study, kernel ridge regression and the multilayer neural network, are applied to this problem. These algorithms are well-established non-linear methods and are good candidates for handling the intrinsic nonlinearities of the problem. In kernel ridge regression, the measure of similarity is encoded in the kernel. On the other hand, in multilayer neural networks, the measure of similarity is learned essentially from data and implicitly given by the mapping onto increasingly many layers. In general, neural networks are more flexible and make less assumptions about the data. However, it comes at the cost of being more difficult to train and regularize.

### 3.1  Kernel Ridge Regression

The most basic algorithm to solve the nonlinear regression problem at hand is kernel ridge regression (cf. Hastie et al., 2001). It uses a quadratic constraint on the norm of $\alpha_i$. As is well known, the solution of the minimization problem

$$\min_{\alpha} \sum_i \left(E^{\text{est}}(x_i) - E_i^{\text{ref}}\right)^2 + \lambda \sum_i \alpha_i^2$$

reads $\alpha = (K + \lambda I)^{-1} E^{\text{ref}}$, where $K$ is the empirical kernel and the input data $x_i$ is either the eigenspectrum of the Coulomb matrix or the vectorized sorted Coulomb matrix.

Expanding the dataset with the randomly generated Coulomb matrices described in Section 2.3 yields a huge dataset that is difficult to handle with standard kernel ridge regression algorithms. Although approximations of the kernel can improve its scalability, random Coulomb matrices can be handled more easily by encoding permutations *directly* into the kernel. We redefine the kernel as

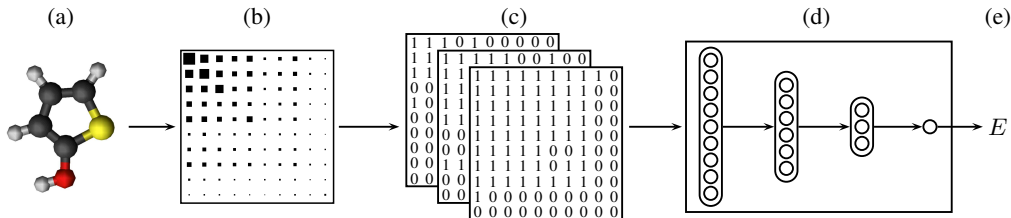

Figure 3: Data flow from the raw molecule to the predicted atomization energy $E$. The molecule (a) is converted to its randomly sorted Coulomb matrix representation (b). The Coulomb matrix is then converted into a suitable sensory input (c) that is fed to the neural network (d). The output of the neural network is then rescaled to the original energy unit (e).

a sum over permutations:

$$\tilde{K}(x_i, x_j) = \frac{1}{2} \sum_{l=1}^{L} \left( K(x_i, P_l(x_j)) + K(P_l(x_i), x_j) \right) \tag{3}$$

where $P_l$ is the $l$-th permutation of atoms corresponding to the $l$-th realization of the random Coulomb matrix and $L$ is the total number of permutations. This sum over multiple permutations has the effect of testing multiple plausible alignments of molecules. Note that the summation can be replaced by a "max" operator in order to focus on correct alignments of molecules and ignore poor alignments.

## 3.2 Multilayer Neural Networks

A main feature of multilayer neural networks is their ability to learn internal representations that potentially make models statistically and computationally more efficient. Unfortunately, the intrinsically non-convex nature of neural networks makes them hard to optimize and regularize in a principled manner. Often, a crucial factor for training neural networks successfully, is to start with a favorable initial conditioning of the learning problem, that is, a good sensory input representation and a proper weights initialization.

Unlike images or speech data, an important amount of label-relevant information is contained within the elements of the Coulomb matrix and not only in their dependencies. For these reasons, taking the real quantities directly as input is likely to lead to a poorly conditioned optimization problem. Instead, we choose to break apart each dimension of the Coulomb matrix $C$ by converting the representation into a three-dimensional tensor of essentially binary predicates as follows:

$$x = \left[ \dots, \tanh\left(\frac{C - \theta}{\theta}\right), \tanh\left(\frac{C}{\theta}\right), \tanh\left(\frac{C + \theta}{\theta}\right), \dots \right] \tag{4}$$

The new representation $x$ is fed as input to the neural network. Note that in the new representation, many elements are constant and can be pruned. In practice, by choosing an appropriate step $\theta$, the dimensionality of the sensory input is kept to tractable levels.

This binarization of the input space improves the conditioning of the learning problem and makes the model more flexible. As we will see in Section 5, learning from this flexible representation requires enough data in order to compensate for the lack of a strong prior and might lead to low performance if this condition is not met. The full data flow from the raw molecule to the predicted atomization energy is depicted in Figure 3.

## 4   Methodology

**Dataset**   As in Rupp et al. (2012), we select a subset of 7165 small molecules extracted from a huge database of nearly one billion small molecules collected by Blum and Reymond (2009). These molecules are composed of a maximum of 23 atoms, a maximum of 7 of them are heavy atoms.

Molecules are converted to a suitable Cartesian coordinates representation using universal force-field method (Rappé et al., 1992) as implemented in the software OpenBabel (Guha et al., 2006). The Coulomb matrices can then be computed from these Cartesian coordinates using Equation 1. Atomization energies are calculated for each molecule and are ranging from $-800$ to $-2000$ kcal/mol. As a result, we have a dataset of 7165 Coulomb matrices of size $23 \times 23$ with their associated one-dimensional labels[2]. Random Coulomb matrices are generated with the noise parameter $\sigma = 1$ (see Equation 2).

**Model validation**  For each learning method we used stratified 5-fold cross validation with identical cross validation folds, where the stratification was done by grouping molecules into groups of five by their energies and then randomly assigning one molecule to each fold, as in Rupp et al. (2012). This sampling reduces the variance of the test error estimator. Each algorithm is optimized for mean squared error. To illustrate how the prediction accuracy changes when increasing the training sample size, each model was trained on 500 to 7000 data points which were sampled identically for the different methods.

**Choice of parameters for kernel ridge regression**  The kernel ridge regression model was trained using a Gaussian kernel ($K_{ij} = \exp[-||x_i - x_j||^2/(2\sigma^2)]$) where $\sigma$ is the kernel width. No further scaling or normalization of the data was done, as the meaningfulness of the data in chemical compound space was to be preserved. A grid search with an inner cross validation was used to determine the hyperparameters for each of the five cross validation folds for each method, namely kernel width $\sigma$ and regularization strength $\lambda$. Grid-searching for optimal hyperparameters can be easily parallelized. The regularization parameter was varied from $10^{-11}$ to $10^1$ on a logarithmic scale and the kernel width was varied from 5 to 81 on a linear scale with a step size of 4. For the eigenspectrum representation the individual folds showed lower regularization parameters ($\lambda_{\mathrm{eig}} = 2.15 \cdot 10^{-10} \pm 0.00$) as compared to the sorted Coulomb representation ($\lambda_{\mathrm{sorted}} = 1.67 \cdot 10^{-7} \pm 0.00$). The optimal kernel width parameters are $\sigma_{\mathrm{eig}} = 41 \pm 6.07$ and $\sigma_{\mathrm{sorted}} = 77 \pm 0.00$. As indicated by the standard deviation 0.00, identical parameters are often chosen for all folds of cross-validation. Training one fold, for one particular set of parameters took approximately 10 seconds. When the algorithm is trained on random Coulomb matrices, we set the number of permutations involved in the kernel to $L = 250$ (see Equation 3) and grid-search hyperparameters over both the "sum" and "max" kernels. Obtained parameters are $\lambda_{\mathrm{random}} = 0.0157 \pm 0.0247$ and $\sigma_{\mathrm{random}} = 74 \pm 4.38$.

**Choice of parameters for the neural network**  We choose a binarization step $\theta = 1$ (see Equation 4). As a result, the neural network takes approximately 1800 inputs. We use two hidden layers composed of 400 and 100 units with sigmoidal activation functions, respectively. Initial weights $W_0$ and learning rates $\gamma$ are chosen as $W_0 \sim \mathcal{N}(0, 1/\sqrt{m})$ and $\gamma = \gamma_0/\sqrt{m}$ where $m$ is the number of input units and $\gamma_0$ is the global learning rate of the network set to $\gamma_0 = 0.01$. The error derivative is backpropagated from layer $l$ to layer $l - 1$ by multiplying it by $\eta = \sqrt{m/n}$ where $m$ and $n$ are the number of input and output units of layer $l$. These choices for $W_0$, $\gamma$ and $\eta$ ensure that the representations at each layer fall into the correct regime of the nonlinearity and that weights in each layer evolve at the correct speed. Inputs and outputs are scaled to have mean 0 and standard deviation 1. We use averaged stochastic gradient descent (ASGD) with minibatches of size 25 for a maximum of 250000 iterations and with ASGD coefficients set so that the neural network remembers approximately 10% of its training history. The training is performed on 90% of the training set and the rest is used for early stopping. Training the neural network takes between one hour and one day on a CPU depending on the sample complexity. When using the random Coulomb matrix representation, the prediction for a new molecule is averaged over 10 different realizations of its associated random Coulomb matrix.

## 5  Results

Cross-validation results for each learning algorithm and representation are shown in Table 1. For the sake of completeness, we also include some baseline results such as the mean predictor (simply predicting the mean of labels in the training set), linear regression, k-nearest neighbors, mixed effects

| Learning algorithm | Molecule representation | MAE | RMSE |
|---|---|---|---|
| Mean predictor | None | $179.02 \pm 0.08$ | $223.92 \pm 0.32$ |
| K-nearest neighbors | Eigenspectrum | $70.72 \pm 2.12$ | $92.49 \pm 2.70$ |
|  | Sorted Coulomb | $71.54 \pm 0.97$ | $95.97 \pm 1.45$ |
| Linear regression | Eigenspectrum | $29.17 \pm 0.35$ | $38.01 \pm 1.11$ |
|  | Sorted Coulomb | $20.72 \pm 0.32$ | $27.22 \pm 0.84$ |
| Mixed effects | Eigenspectrum | $10.50 \pm 0.48$ | $20.38 \pm 9.29$ |
|  | Sorted Coulomb | $8.5 \pm 0.45$ | $12.16 \pm 0.95$ |
| Gaussian support vector regression | Eigenspectrum | $10.78 \pm 0.58$ | $19.47 \pm 9.46$ |
|  | Sorted Coulomb | $8.06 \pm 0.38$ | $12.59 \pm 2.17$ |
| Gaussian kernel ridge regression | Eigenspectrum | $11.39 \pm 0.81$ | $16.01 \pm 1.71$ |
|  | Sorted Coulomb | $8.72 \pm 0.40$ | $12.59 \pm 1.35$ |
|  | Random Coulomb | $7.79 \pm 0.42$ | $11.40 \pm 1.11$ |
| Multilayer neural network | Eigenspectrum | $14.08 \pm 0.29$ | $20.29 \pm 0.73$ |
|  | Sorted Coulomb | $11.82 \pm 0.45$ | $16.01 \pm 0.81$ |
|  | Random Coulomb | $3.51 \pm 0.13$ | $5.96 \pm 0.48$ |

Table 1: Prediction errors in terms of mean absolute error (MAE) and root mean square error (RMSE) for several algorithms and types of representations. Linear regression and k-nearest neighbors are inaccurate compared to the more refined kernel methods and multilayer neural network. The multilayer neural network performance varies considerably depending on the type of representation but sets the lowest error in our study on the random Coulomb representation.

models (Pinheiro and Bates, 2000; Fazli et al., 2011) and kernel support vector regression (Smola and Schölkopf, 2004). Linear regression and k-nearest neighbors are clearly off-the-mark compared to the other more sophisticated models such as mixed effects models, kernel methods and multilayer neural networks.

While results for kernel algorithms are similar, they all differ considerably from those obtained with the multilayer neural network. In particular, we can observe that they are performing reasonably well with all types of representation while the multilayer neural network performance is highly dependent on the representation fed as input.

More specifically, the multilayer neural network tends to perform better as the input representation gets richer (as the total amount of information in the input distribution increases), suggesting that the lack of a strong inbuilt prior in the neural network must be compensated by a large amount of data. The neural network performs best with random Coulomb matrices that are intrinsically the richest representation as a whole distribution over Coulomb matrices is associated to each molecule.

A similar phenomenon can be observed from the learning curves in Figure 4. As the training data increases, the error for Gaussian kernel ridge regression decreases slowly while the neural network can take greater advantage from this additional data.

## 6   Conclusion

Predicting molecular energies quickly and accurately across the chemical compound space (CCS) is an important problem as the quantum-mechanical calculations are typically taking days and do not scale well to more complex systems. Supervised statistical learning is a natural candidate for solving this problem as it encourages computational units to focus on solving the problem of interest rather than solving the more general Schrödinger equation.

In this paper, we have developed further the learning-from-scratch approach initiated by Rupp et al. (2012) and provided a deeper understanding of some of the ingredients for learning a successful mapping between raw molecular geometries and atomization energies. Our results suggest the importance of having flexible priors (in our case, a multilayer network) and lots of data (generated artificially by exploiting symmetries of the Coulomb matrix). Our work improves the state-of-the-art on this dataset by a factor of almost three. From a reference MAE of 9.9 kcal/mol (Rupp et al.,

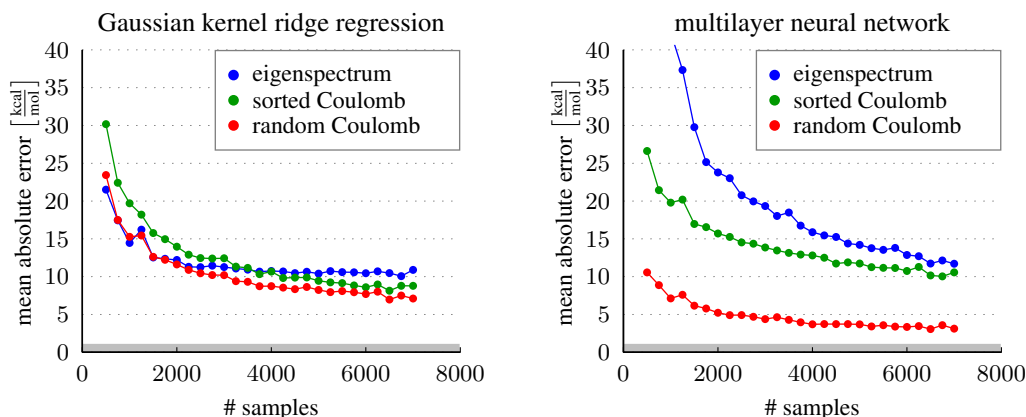

Figure 4: Learning curves for Gaussian kernel ridge regression and the multilayer neural network. Results for kernel ridge regression are more invariant to the representation and to the number of samples than for the multilayer neural network. The gray area at the bottom of the plot indicates the level at which the prediction is considered to be "chemically accurate".

2012), we went down to a MAE of 3.51 kcal/mol, which is considerably closer to the 1 kcal/mol required for chemical accuracy.

Many open problems remain that makes quantum chemistry an attractive challenge for Machine Learning: (1) Are there fundamental modeling limits of the statistical learning approach for quantum chemistry applications or is it rather a matter of producing more training data? (2) The training data can be considered noise free. Thus, are there better ML models for the noise free case while regularizing away the intrinsic problem complexity to keep the ML model small? (3) Can better representations be devised with inbuilt invariance properties (e.g. Tangent Distance, Simard et al., 1996), harvesting physical prior knowledge? (4) How can we extract physics insights on quantum mechanics from the trained nonlinear ML prediction models?

### Acknowledgments

This work is supported by the World Class University Program through the National Research Foundation of Korea funded by the Ministry of Education, Science, and Technology, under Grant R31-10008, and the FP7 program of the European Community (Marie Curie IEF 273039). This research used resources of the Argonne Leadership Computing Facility at Argonne National Laboratory, which is supported by the Office of Science of the U.S. DOE under Contract No. DE-AC02-06CH11357. This research is supported, in part, by the Natural Sciences and Engineering Research Council of Canada. The authors also thank Márton Danóczy for preliminary work and useful discussions.

## Footnotes

* Electronic address: gregoire.montavon@tu-berlin.de

† Electronic address: klaus-robert.mueller@tu-berlin.de

[1] This approach has already been applied in multiple domains such as natural language processing (Collobert et al., 2011) or speech recognition (Jaitly and Hinton, 2011).

[2]The dataset is available at http://www.quantum-machine.org.

## References

Roman M. Balabin and Ekaterina I. Lomakina. Neural network approach to quantum-chemistry data: Accurate prediction of density functional theory energies. *Journal of Chemical Physics*, 131(7):074104, 2009.

Roman M. Balabin and Ekaterina I. Lomakina. Support vector machine regression (LS-SVM)—an alternative to artificial neural networks (ANNs) for the analysis of quantum chemistry data? *Physical Chemistry Chemical Physics*, 13(24):11710–11718, 2011.

Pierre Baldi, Klaus-Robert Müller, and Gisbert Schneider. Editorial: Charting chemical space: Challenges and opportunities for artificial intelligence and machine learning. *Molecular Informatics*, 30(9):751–751, 2011.

Albert P. Bartók, Mike C. Payne, Risi Kondor, and Gábor Csányi. Gaussian approximation potentials: The accuracy of quantum mechanics, without the electrons. *Phys. Rev. Lett.*, 104(13): 136403, 2010.

Jörg Behler. Neural network potential-energy surfaces in chemistry: a tool for large-scale simulations. *Physical Chemistry Chemical Physics*, 13(40):17930–17955, 2011.

Lorenz C. Blum and Jean-Louis Reymond. 970 million druglike small molecules for virtual screening in the chemical universe database GDB-13. *Journal of the American Chemical Society*, 131 (25):8732–8733, 2009.

Dan Claudiu Ciresan, Ueli Meier, Luca Maria Gambardella, and Jürgen Schmidhuber. Deep, big, simple neural nets for handwritten digit recognition. *Neural Computation*, 22(12):3207–3220, 2010.

Ronan Collobert, Jason Weston, Léon Bottou, Michael Karlen, Koray Kavukcuoglu, and Pavel Kuksa. Natural language processing (almost) from scratch. *Journal of Machine Learning Research*, 12:2493–2537, 2011.

Dennis DeCoste and Bernhard Schölkopf. Training invariant support vector machines. *Machine Learning*, 46(1–3):161–190, 2002.

Siamac Fazli, Márton Danóczy, Jürg Schelldorfer, and Klaus-Robert Müller. $\ell_1$-penalized linear mixed-effects models for high dimensional data with application to BCI. *NeuroImage*, 56(4): 2100–2108, 2011.

Rajarshi Guha, Michael T. Howard, Geoffrey R. Hutchison, Peter Murray-Rust, Henry Rzepa, Christoph Steinbeck, Jörg Wegner, and Egon L. Willighagen. The blue obelisk, interoperability in chemical informatics. *Journal of Chemical Information and Modeling*, 46(3):991–998, 2006.

Trevor Hastie, Robert Tibshirani, and Jerome Friedman. *The Elements of Statistical Learning*. Springer Series in Statistics. Springer New York Inc., 2001.

Geoffroy Hautier, Christopher C. Fisher, Anubhav Jain, Tim Mueller, and Gerbrand Ceder. Finding nature's missing ternary oxide compounds using machine learning and density functional theory. *Chemistry of Materials*, 22(12):3762–3767, 2010.

Navdeep Jaitly and Geoffrey E. Hinton. Learning a better representation of speech soundwaves using restricted Boltzmann machines. In *ICASSP*, pages 5884–5887, 2011.

Yann LeCun, Léon Bottou, Yoshua Bengio, and Patrick Haffner. Gradient-based learning applied to document recognition. *Proceedings of the IEEE*, 86(11):2278–2324, 1998.

Sönke Lorenz, Axel Groß, and Matthias Scheffler. Representing high-dimensional potential-energy surfaces for reactions at surfaces by neural networks. *Chemical Physics Letters*, 395(4–6):210–215, 2004.

Sergei Manzhos and Tucker Carrington. A random-sampling high dimensional model representation neural network for building potential energy surfaces. *J. Chem. Phys.*, 125:084109, 2006.

José C. Pinheiro and Douglas M. Bates. *Mixed-Effects Models in S and S-Plus*. Springer, New York, 2000.

Anthony K. Rappé, Carla J. Casewit, K. S. Colwell, William A. Goddard, and W. M. Skiff. UFF, a full periodic table force field for molecular mechanics and molecular dynamics simulations. *Journal of the American Chemical Society*, 114(25):10024–10035, 1992.

Matthias Rupp, Alexandre Tkatchenko, Klaus-Robert Müller, and O. Anatole von Lilienfeld. Fast and accurate modeling of molecular atomization energies with machine learning. *Phys. Rev. Lett.*, 108(5):058301, 2012.

Patrice Simard, Yann LeCun, John S. Denker, and Bernard Victorri. Transformation invariance in pattern recognition: Tangent distance and tangent propagation. In *Neural Networks: Tricks of the Trade*, pages 239–27, 1996.

Alex J. Smola and Bernd Schölkopf. A tutorial on support vector regression. *Statistics and computing*, 14(3):199–222, 2004.

Roberto Todeschini and Viviana Consonni. *Handbook of Molecular Descriptors*. Wiley-VCH, Weinheim, Germany, second edition, 2009.

O Anatole Von Lilienfeld and Mark E. Tuckerman. Molecular grand-canonical ensemble density functional theory and exploration of chemical space. *The Journal of chemical physics*, 125(15): 154104, 2006.

